# Learning Body Pose via Specialized Maps

**Rómer Rosales**
Department of Computer Science
Boston University, Boston, MA 02215
*rrosales@cs.bu.edu*

**Stan Sclaroff**
Department of Computer Science
Boston University, Boston, MA 02215
*sclaroff@cs.bu.edu*

## Abstract

A nonlinear supervised learning model, the Specialized Mappings Architecture (SMA), is described and applied to the estimation of human body pose from monocular images. The SMA consists of several specialized forward mapping functions and an inverse mapping function. Each specialized function maps certain domains of the input space (image features) onto the output space (body pose parameters). The key algorithmic problems faced are those of learning the specialized domains and mapping functions in an optimal way, as well as performing inference given inputs and knowledge of the inverse function. Solutions to these problems employ the EM algorithm and alternating choices of conditional independence assumptions. Performance of the approach is evaluated with synthetic and real video sequences of human motion.

## 1 Introduction

In everyday life, humans can easily estimate body part locations (body pose) from relatively low-resolution images of the projected 3D world (*e.g.,* when viewing a photograph or a video). However, body pose estimation is a very difficult computer vision problem. It is believed that humans employ extensive prior knowledge about human body structure and motion in this task [10]. Assuming this, we consider how a computer might learn the underlying structure and thereby infer body pose.

In computer vision, this task is usually posed as a *tracking* problem. Typically, models comprised of 2D or 3D geometric primitives are designed for tracking a specific articulated body [13, 5, 2, 15]. At each frame, these models are fitted to the image to optimize some cost function. Careful manual placement of the model on the first frame is required, and tracking in subsequent frames tends to be sensitive to errors in initialization and numerical drift. Generally, these systems cannot recover from tracking errors in the middle of a sequence. To address these weaknesses, more complex dynamic models have been proposed [14, 13, 9]; these methods learn a prior over some specific motion (such as walking). This strong prior however, substantially limits the generality of the motions that can be tracked.

Departing from the aforementioned tracking paradigm, in [8] a Gaussian probability model was learned for short human motion sequences. In [17] dynamic programming was used to calculate the best global labeling according to the learned joint probability density function of the position and velocity of body features. Still, in these approaches, the joint locations, correspondences, or model initialization must be provided by hand. In [1], the manifold of human body dynamics was modeled via a hidden Markov model and learned via entropic minimization. In all of these approaches models were learned. Although the approach presented here can be used to model dynamics, we argue that when general human motion dynamics are intended to be learned, the amount of training data, model complexity, and computational resources required are impractical. As a consequence, models with large priors towards specific motions (*e.g.,* walking) are generated. In this paper we describe a non-linear supervised learning algorithm, the Specialized Maps Architecture (SMA), for recovering articulated body pose from single monocular images. This approach avoids the need for initialization and tracking *per se*, and reduces the above mentioned disadvantages.

## 2   Specialized Maps

There at least two key characteristics of the problem we are trying to solve which make it different from other supervised learning problems. First, we have access to the *inverse* map. We are trying to learn unknown probabilistic maps from inputs to outputs space, but we have access to the map (in general probabilistic) from outputs to inputs. In our pose estimation problem, it is easy to see how we can artificially, using computer graphics (CG), produce *some* visual features (*e.g.,* body silhouettes) given joint positions[1]. Second, it is one-to-many: one input can be associated with more than one output. Features obtained from silhouettes (and many other visual features) are ambiguous. Consider an occluded arm, or the reflective ambiguity generated by symmetric poses. This last observation precludes the use of standard algorithms for supervised learning that fit a single mapping function to the data.

Given input and output spaces $\Re^c$ and $\Re^t$, and the *inverse* function $\zeta : \Re^t \to \Re^c$, we describe a solution for these supervised learning problems. Our approach consists in generating a series of $m$ functions $\phi_k : \Re^c \to \Re^t$. Each of these functions is specialized to map only certain inputs (for a specialized sub-domain) better than others. For example, each sub-domain can be a region of the input space. However, the specialized sub-domain of $\phi_k$ can be more general than just a connected region in the input space.

Several other learning models use a similar concept of fitting surfaces to the observed data by splitting the input space into several regions and approximating simpler functions in these regions (*e.g.,* [11, 7, 6]). However, in these approaches, the inverse map is not incorporated in the estimation algorithm because it is not considered in the problem definition and the forward model is usually more complex, making inference and learning more difficult.

The key algorithmic problems are that of estimating the specialized domains and functions in an optimal way (taking into account the form of the specialized functions), and using the knowledge of the inverse function to formulate efficient infer-

ence and learning algorithms. We propose to determine the specialized domains and functions using an approximate EM algorithm and to perform inference using, in an alternating fashion, the conditional independence assumptions specified by the forward and inverse models. Fig. 1(a) illustrates a learned forward model.

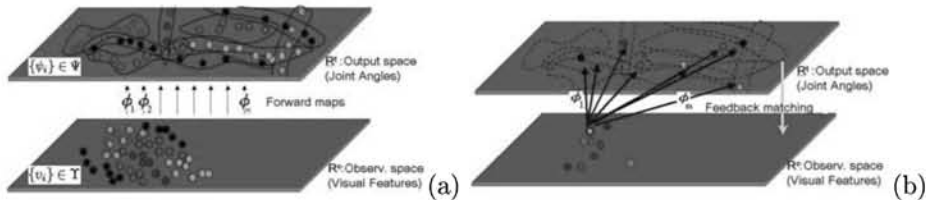

Figure 1: SMA diagram illustrating (a) an already learned SMA model with $m$ specialized functions mapping subsets of the training data, each subset is drawn with a different color (at initializations, coloring is random) and (b) the mean-output inference process in which a given observation is mapped by all the specialized functions, and then a feedback matching step, using $\zeta$, is performed to choose the best of the $m$ estimates.

## 3 Probabilistic Model

Let the training sets of output-input observations be $\mathbf{\Psi} = \{\psi_1, ..., \psi_N\}$, and $\mathbf{\Upsilon} = \{v_1, ..., v_N\}$ respectively. We will use $\mathbf{z}_i = (\psi_i, v_i)$ to define the given output-input training pair, and $\mathcal{Z} = \{\mathbf{z}_1, ..., \mathbf{z}_N\}$ as our observed training set.

We introduce the unobserved random variable $\mathbf{y} = (y_1, ..., y_n)$. In our model any $y_i$ has domain the discrete set $\mathcal{C} = \{1, ..., M\}$ of labels for the specialized functions, and can be thought as the function number used to map data point $i$; thus $M$ is the number of specialized mapping functions. Our model uses parameters $\theta = (\theta_1, ..., \theta_M, \lambda)$, $\theta_k$ represents the parameters of the mapping function $k$; $\lambda = (\lambda_1, ..., \lambda_M)$, where $\lambda_k$ represents $P(y_i = k|\theta)$: the prior probability that mapping function with label $i$ will be used to map an unknown point. As an example, $P(y_i|\mathbf{z}_i, \theta)$ represents the probability that function number $y_i$ generated data point number $i$.

Using Bayes' rule and assuming independence of observations given $\theta$, we have the log-probability of our data given the model $\log p(\mathcal{Z}|\theta)$, which we want to maximize:

$$\arg\max_\theta \sum_i \log \sum_k p(\psi_i|v_i, y_i = k, \theta) P(y_i = k|\theta) p(v_i), \tag{1}$$

where we used the independence assumption $p(v|\theta) = p(v)$. This is also equivalent to maximizing the conditional likelihood of the model.

Because of the log-sum encountered, this problem is intractable in general. However, there exist practical approximate optimization procedures, one of them is Expectation Maximization (EM) [3, 4, 12].

### 3.1 Learning

The EM algorithm is well known, therefore here we only provide the derivations specific to SMA's. The E-step consists of finding $P(\mathbf{y} = \mathbf{k}|\mathbf{z}, \theta) = \tilde{P}(\mathbf{y})$. Note that the variables $y_i$ are assumed independent (given $\mathbf{z}_i$). Thus, factorizing $\tilde{P}(\mathbf{y})$:

$$\tilde{P}(\mathbf{y}) = \prod_i \tilde{P}^{(t)}(y_i) = \prod_i [(\lambda_{y_i} p(\psi_i | v_i, y_i, \theta)) / (\sum_{k \in \mathcal{C}} \lambda_k p(\psi_i | v_i, y_i = k, \theta))] \quad (2)$$

However, $p(\psi_i | v_i, y_i = k, \theta)$ is still undefined. For the implementation described in this paper we use $\mathcal{N}(\psi_i; \phi_k(v_i, \theta_k), \Sigma_k)$, where $\theta_k$ are the parameters of the $k$-th specialized function, and $\Sigma_k$ the error covariance of the specialized function $k$. One way to interpret this choice is to think that the error cost in estimating $\psi$ once we know the specialized function to use, is a Gaussian distribution with mean the output of the specialized function and some covariance which is map dependent. This also led to tractable further derivations. Other choices were given in [16].

The M-step consists of finding $\theta^{(t)} = \arg\max_\theta E_{\tilde{P}^{(t)}}[\log p(\mathcal{Z}, \mathbf{y}|\theta)]$. In our case we can show that this is equivalent to finding:

$$\arg\min_\theta \sum_i \sum_k \tilde{P}^{(t)}(y_i = k)(\psi_i - \phi_k(v_i, \theta_k))^\top \Sigma_k^{-1} (\mathbf{z}_i - \phi_k(\mathbf{z}_i, \theta_k)). \quad (3)$$

This gives the following update rules for $\lambda_k$ and $\Sigma_k$ (where Lagrange multipliers were used to incorporate the constraint that the sum of the $\lambda_k$'s is 1.

$$\lambda_k = \frac{1}{n} \sum_i P(y_i = k|\mathbf{z}_i, \theta) \quad (4)$$

$$\Sigma_k = \sum_i \tilde{P}^{(t)}(y_i = k)(\psi_i - \phi_k(v_i, \theta_k))(\psi_i - \phi_k(v_i, \theta_k))^\top / \sum_i \tilde{P}^{(t)}(y_i = k) \quad (5)$$

In keeping the formulation general, we have not defined the form of the specialized functions $\phi_k$. Whether or not we can find a closed form solution for the update of $\theta_k$ depends on the form of $\phi_k$. For example if $\phi_k$ is a non-linear function, we may have to use iterative optimization to find $\theta_k^{(t)}$. In case $\phi_k$ yield a quadratic form, then a closed form update exists. However, in general we have:

$$\frac{\partial E}{\partial \theta_k} = \sum_i \tilde{P}_i^{(t)}(y_i = k)[(\frac{\partial}{\partial \theta_k} \phi_k(v_i, \theta_k))^\top \Sigma_k^{-1} (\psi_i - \phi_k(v_i, \theta_k))], \quad (6)$$

In our experiments, $\phi_k$ is a 1-hidden layer perceptron. Thus, the M-step is an approximate, iterative optimization procedure.

## 4 Inference

Once learning is accomplished, each specialized function maps (with different levels of accuracy) the input space. We can formally state the inference process as that of maximum-a-posteriori (MAP) estimation where we are interested in finding the most likely output $\mathbf{h}$ given an input configuration $\mathbf{x}$:

$$\mathbf{h}^* = \arg\max_\mathbf{h} p(\mathbf{h}|\mathbf{x}) = \arg\max_\mathbf{h} \sum_y p(\mathbf{h}|y, \mathbf{x}) P(y), \quad (7)$$

Any further treatment depends on the properties of the probability distributions involved. If $p(\mathbf{h}|\mathbf{x}, y) = \mathcal{N}(\mathbf{h}; \phi_y(\mathbf{x}), \Sigma_y)$, the MAP estimate involves finding the maximum in a mixture of Gaussians. However, no closed form solution exists and moreover, we have not incorporated the potentially useful knowledge of the inverse function $\zeta$.

## 4.1 MAP by Using the Inverse Function $\zeta$

The access to a forward kinematics function $\zeta$ (called here the inverse function) allows to formulate a different inference algorithm. We are again interested in finding an optimal $\mathbf{h}^*$ given an input $\mathbf{x}$ (*e.g.*, an optimal body pose given features taken from an image). This can be formulated as:

$$\mathbf{h}^* = \arg\max_{\mathbf{h}} p(\mathbf{h}|\mathbf{x}) = \arg\max_{\mathbf{h}} p(\mathbf{x}|\mathbf{h}) \sum_y p(\mathbf{h}|y,\mathbf{x})P(y), \tag{8}$$

simply by Bayes' rule, and marginalizing over all variables except $\mathbf{h}$. Note that we have made the distribution $p(\mathbf{x}|\mathbf{h})$ appear in the solution. This is important because we can know use our knowledge of $\zeta$ to define this distribution. This solution is completely general within our architecture, we did not make any assumptions on the form of the distributions or algorithms used.

## 5 Approximate Inference using $\zeta$

Let us assume that we can approximate $\sum_y p(\mathbf{h}|y,\mathbf{x})P(y)$ by a set of samples generated according to $p(\mathbf{h}|y,\mathbf{x})P(y)$ and a kernel function $K(\mathbf{h},\mathbf{h}_s)$. Denote the set of samples $\mathcal{H}_{Spl} = \{\mathbf{h}_s\}_{s=1\ldots S}$. An approximate to $\sum_y p(\mathbf{h}|y,\mathbf{x})P(y)$ is formally built by $\frac{1}{S}\sum_{s=1}^S K(\mathbf{h},\mathbf{h}_s)$, with the normalizing condition $\int K(\mathbf{h},\mathbf{h}_s)d\mathbf{h} = 1$ for any given $\mathbf{h}_s$.

We will consider two simple forms of $K$. If $K(\mathbf{h},\mathbf{h}_s) = \delta(\mathbf{h} - \mathbf{h}_s)$, we have: $\hat{\mathbf{h}} = \arg\max_{\mathbf{h}} p(\mathbf{x}|\mathbf{h}) \sum_{s=1}^S \delta(\mathbf{h} - \mathbf{h}_s)$.

After some simple manipulations, this can be reduced to the following equivalent discrete optimization problem whose goal is to find the most likely sample $s^*$:

$$s^* = \arg\max_s p(\mathbf{x}|\mathbf{h}_s) = \arg\min_s (\mathbf{x} - \zeta(\mathbf{h}_s))^\top \Sigma_\zeta (\mathbf{x} - \zeta(\mathbf{h}_s)), \tag{9}$$

where the last equivalence used the assumption $p(\mathbf{x}|\mathbf{h}) = \mathcal{N}(\mathbf{x};\zeta(\mathbf{h}),\Sigma_\zeta)$.

If $K(\mathbf{h},\mathbf{h}_s) = \mathcal{N}(\mathbf{h};\mathbf{h}_s,\Sigma_{Spl})$, we have: $\hat{\mathbf{h}} = \arg\max_{\mathbf{h}} p(\mathbf{x}|\mathbf{h}) \sum_{s=1}^S \mathcal{N}(\mathbf{h};\mathbf{h}_s,\Sigma_{Spl})$. This case is hard to use in practice, because contrary to the case above (Eq. 9), in general, there is no guarantee that the optimal $\mathbf{h}$ is among the samples.

### 5.1 A Deterministic Approximation based on the Functions Mean Output

The structure of the inference in SMA, and the choice of probabilities $p(\mathbf{h}|\mathbf{x},y)$ allows us to construct a newer approximation that is considerably less expensive to compute, and it is deterministic. Intuitively they idea consists of *asking* each of the specialized functions $\phi_k$ what their most likely estimate for $\mathbf{h}$ is, given the observed input $\mathbf{x}$. The *opinions* of each of these specialized functions are then evaluated using our distribution $p(\mathbf{x}|\mathbf{h})$ similar to the above sampling method.

This can be justified by the observation that the probability of the mean is maximal in a Gaussian distribution. Thus by considering the means $\phi_k(\mathbf{x})$, we would be considering the most likely output of each specialized function. Of course, in many cases this approximation could be very far from the best solution, for example when

the uncertainty in the function estimate is relatively high relative to the difference between means.

We use Fig. 1(b) to illustrate the mean-output (MO) approximate inference process. When generating an estimate of body pose, denoted $\hat{\mathbf{h}}$, given an input $\mathbf{x}$ (the gray point with a dark contour in the lower plane), the SMA generates a series of output hypotheses $\mathcal{H}_\phi = \{\mathbf{h}_k^\phi\}_k$ obtained using $\mathbf{h}_k = \phi_k(\mathbf{x})$, with $k \in \mathcal{C}$ (illustrated by each of the points pointed to by the arrows).

Given the set $\mathcal{H}_\phi$, the most accurate hypothesis under the mean-output criteria is the one that minimizes the function:

$$k^* = \arg\min_k (\mathbf{x} - \zeta(\mathbf{h}_k^\phi))^\top \Sigma_\zeta (\mathbf{x} - \zeta(\mathbf{h}_k^\phi)), \qquad (10)$$

where in the last equation we have assumed $p(\mathbf{x}|\mathbf{h})$ is Gaussian.

## 5.2 Bayesian Inference

Note that in many cases, there may not be any need to simply provide a *point* estimate, in terms of a most likely output $\mathbf{h}$. In fact we could instead use the whole distribution found in the inference process. We can show that using the above choices for $K$ we can respectively obtain.

$$p(\mathbf{h}|\mathbf{x}) = \frac{1}{S} \sum_{s=1}^{S} \mathcal{N}(\mathbf{x}; \zeta(\mathbf{h}_s), \Sigma_\zeta), \qquad (11)$$

$$p(\mathbf{h}|\mathbf{x}) = \mathcal{N}(\mathbf{h}; \mathbf{h}_s, \Sigma_{Spl}) \sum_{s=1}^{S} \mathcal{N}(\mathbf{x}; \zeta(\mathbf{h}), \Sigma_\zeta). \qquad (12)$$

# 6  Experiments

The described architecture was tested using a computer graphics rendering as our $\zeta$ inverse function. The training data set consisted of approx. 7,000 frames of human body poses obtained through motion capture. The output consisted of 20 2D marker positions (*i.e.*, 3D markers projected to the image plane using a perspective model) but linearly encoded by 8 real values using Principal Component Analysis (PCA). The input (visual features) consisted of 7 real-valued Hu moments computed on synthetically generated silhouettes of the articulated figure. For training/testing we generated 120,000 data points: our 3D poses from motion capture were projected to 16 views along the view-sphere equator. We took 8,000 for training and the rest for testing. The only free parameter in this test, related to the given SMA, was the number of specialized functions used; this was set to 15. For this, several model selection approaches could be used instead. Due to space limitations, in this paper we show results using the mean-output inference algorithm only, readers are referred to http://cs-people.bu.edu/rrosales/SMABodyInference where inference using multiple samples is shown.

Fig. 2(left) shows the reconstruction obtained in several single images coming from three different artificial sequences. The agreement between reconstruction and observation is easy to perceive for all sequences. Note that for self-occluding configurations, reconstruction is harder, but still the estimate is close to ground-truth. No

human intervention nor pose initialization was required. For quantitative results, Fig. 2(right) shows the average marker error and variance per body orientation in percentage of body height. Note that the error is bigger for orientations closer to 0 and $\pi$ radians. This intuitively agrees with the notion that at those angles (side-views), there is less visibility of the body parts. We consider this performance promising, given the complexity of the task and the simplicity of the approach. By choosing poses at random from training set, the RMSE was 17% of body height. In related work, quantitative performance have been usually ignored, in part due to the lack of ground-truth and standard evaluation data sets.

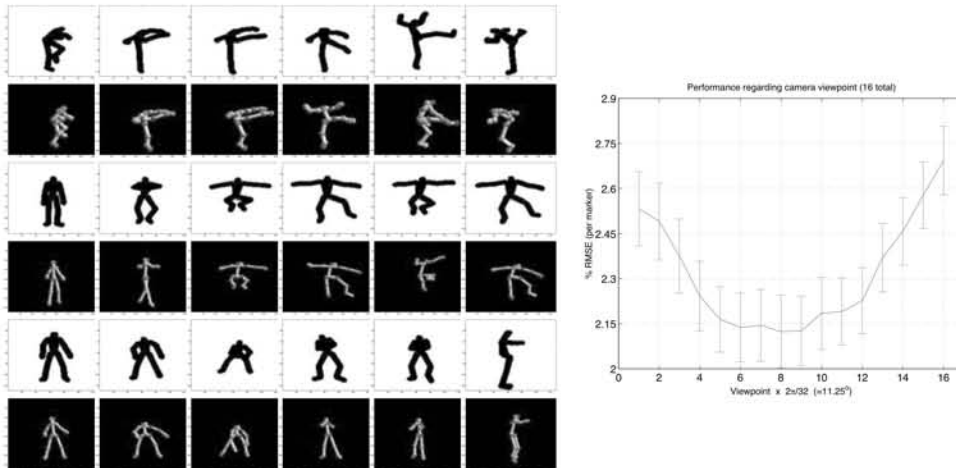

Figure 2: Left: Example reconstruction of several test sequences with CG-generated silhouettes. Each set consists of input images and reconstruction (every 5th frame). Right: Marker root-mean-square-error and variance per camera viewpoint (every $2\pi/32$ rads.). Units are percentage of body height. Approx. 110,000 test poses were used.

### 6.1 Experiments using Real Visual Cues

Fig. 3 shows examples of system performance with real segmented visual data, obtained from observing a human subject. Reconstruction for several relatively complex sequences are shown. Note that even though the characteristics of the segmented body differ from the ones used for training, good performance is still achieved. Most reconstructions are visually close to what can be thought as the right pose reconstruction. Body orientation is also generally accurate.

## 7 Conclusion

In this paper, we have proposed the Specialized Mappings Architecture (SMA). A learning algorithm was developed for this architecture using ideas from ML estimation and latent variable models. Inference was based on the possibility of alternatively use different sets of conditional independence assumptions specified by the forward and inverse models. The incorporation of the inverse function in the model allows for simpler forward models. For example the inverse function is an architectural alternative to the gating networks of Mixture of Experts [11]. SMA advantages for body pose estimation include: no iterative methods for inference are used, the

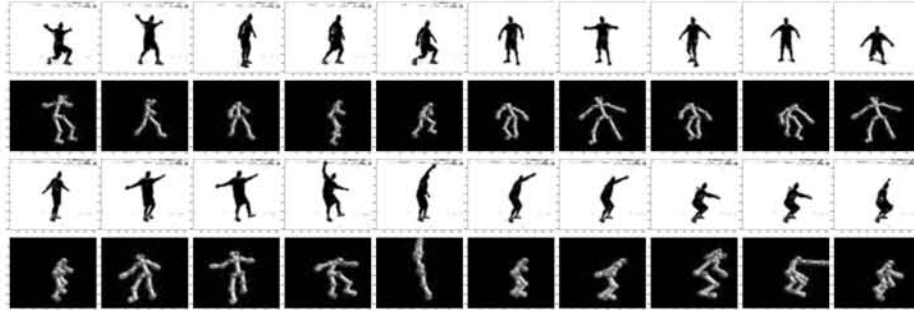

Figure 3: Reconstruction obtained from observing a human subject (every 10th frame).

algorithm for inference runs in constant time and scales only linearly $O(M)$ with respect to the number of specialized functions $M$; manual initialization is not required; compared to approaches that learn dynamical models, the requirements for data are much smaller, and also large priors to specific motions are prevented thus improving generalization capabilities.

## Footnotes

[1]Thus, $\zeta$ is a computer graphics rendering, in general called forward kinematics

## References

[1] M. Brand. Shadow puppetry. In *ICCV*, 1999.

[2] C. Bregler. Tracking people with twists and exponential maps. In *CVPR*, 1998.

[3] I. Csiszar and G. Tusnady. Information geometry and alternating minimization procedures. *Statistics and Decisions*, 1:205–237, 1984.

[4] A. Dempster, N. Laird, and D. Rubin. Maximum likelihood estimation from incomplete data. *Journal of the Royal Statistical Society (B)*, 39(1), 1977.

[5] J. Deutscher, A. Blake, and I. Reid. Articulated body motion capture by annealed particle filtering. In *CVPR*, 2000.

[6] J.H. Friedman. Multivatiate adaptive regression splines. *The Annals of Statistics*, 19,1-141, 1991.

[7] G. Hinton, B. Sallans, and Z. Ghahramani. A hierarchical community of experts. *Learning in Graphical Models, M. Jordan (editor)*, 1998.

[8] N. Howe, M. Leventon, and B. Freeman. Bayesian reconstruction of 3d human motion from single-camera video. In *NIPS-12*, 2000.

[9] M. Isard and A. Blake. Contour tracking by stochastic propagation of conditional density. In *ECCV*, 1996.

[10] G. Johansson. Visual perception of biological motion and a model for its analysis. *Perception and Psychophysics*, 14(2): 210-211, 1973.

[11] M. I. Jordan and R. A. Jacobs. Hierarchical mixtures of experts and the EM algorithm. *Neural Computation*, 6, 181-214, 1994.

[12] R. Neal and G. Hinton. A view of the em algorithm that justifies incremental, sparse, and other variants. *Learning in Graphical Models, M. Jordan (editor)*, 1998.

[13] Dirk Ormoneit, Hedvig Sidenbladh, Michael J. Black, and Trevor Hastie. Learning and tracking cyclic human motion. In *NIPS-13*, 2001.

[14] Vladimir Pavlović, James M. Rehg, and John MacCormick. Learning switching linear models of human motion. In *NIPS-13*, 2001.

[15] J. M. Regh and T. Kanade. Model-based tracking of self-occluding articulated objects. In *ICCV*, 1995.

[16] R. Rosales and S. Sclaroff. Specialized mappings and the estimation of body pose from a single image. In *IEEE Human Motion Workshop*, 2000.

[17] Y. Song, Xiaoling Feng, and P. Perona. Towards detection of human motion. In *CVPR*, 2000.
